# A Posteriori Error Bounds for Joint Matrix Decomposition Problems

**Nicolò Colombo**
Department of Statistical Science
University College London
nicolo.colombo@ucl.ac.uk

**Nikos Vlassis**
Adobe Research
San Jose, CA
vlassis@adobe.com

## Abstract

Joint matrix triangularization is often used for estimating the joint eigenstructure of a set M of matrices, with applications in signal processing and machine learning. We consider the problem of approximate joint matrix triangularization when the matrices in M are jointly diagonalizable and real, but we only observe a set M' of noise perturbed versions of the matrices in M. Our main result is a first-order upper bound on the distance between any approximate joint triangularizer of the matrices in M' and any exact joint triangularizer of the matrices in M. The bound depends only on the observable matrices in M' and the noise level. In particular, it does not depend on optimization specific properties of the triangularizer, such as its proximity to critical points, that are typical of existing bounds in the literature. To our knowledge, this is the first a posteriori bound for joint matrix decomposition. We demonstrate the bound on synthetic data for which the ground truth is known.

## 1 Introduction

Joint matrix decomposition problems appear frequently in signal processing and machine learning, with notable applications in independent component analysis [7], canonical correlation analysis [20], and latent variable model estimation [5, 4]. Most of these applications reduce to some instance of a tensor decomposition problem, and the growing interest in joint matrix decomposition is largely motivated by such reductions. In particular, in the past decade several 'matricization' methods have been proposed for factorizing tensors by computing the joint decomposition of sets of matrices extracted from slices of the tensor (see, e.g., [10, 22, 17, 8]).

In this work we address a standard joint matrix decomposition problem, in which we assume a set of *jointly diagonalizable* ground-truth (unobserved) matrices

$$\mathcal{M}_\circ = \{M_n = V\text{diag}([\Lambda_{n1}, \ldots, \Lambda_{nd}])V^{-1}, \ V \in \mathbb{R}^{d \times d}, \ \Lambda \in \mathbb{R}^{N \times d}\}_{n=1}^N, \qquad (1)$$

which have been corrupted by noise and we observe their noisy versions:

$$\mathcal{M}_\sigma = \{\hat{M}_n = M_n + \sigma R_n, \ M_n \in \mathcal{M}_\circ, \ R_n \in \mathbb{R}^{d \times d}, \ \|R_n\| \le 1\}_{n=1}^N. \qquad (2)$$

The matrices $\hat{M}_n \in \mathcal{M}_\sigma$ are the only observed quantities. The scalar $\sigma > 0$ is the noise level, and the matrices $R_n$ are arbitrary noise matrices with Frobenius norm $\|R_n\| \le 1$. The key problem is to estimate from the observed matrices in $\mathcal{M}_\sigma$ the joint eigenstructure $V, \Lambda$ of the ground-truth matrices in $\mathcal{M}_\circ$. One way to address this estimation problem is by trying to approximately jointly diagonalize the observed matrices in $\mathcal{M}_\sigma$, for instance by directly searching for an invertible matrix that approximates $V$ in (1). This approach is known as nonorthogonal joint diagonalization [23, 15, 18], and is often motivated by applications that reduce to nonsymmetric CP tensor decomposition (see, e.g., [20]).

An alternative approach to the above estimation problem (in the general case of nonorthogonal $V$) is via *joint triangularization*, also known as joint or simultaneous Schur decomposition [1, 13, 11, 12, 22, 8]. Under mild conditions [14], the ground-truth matrices in $\mathcal{M}_\circ$ can be jointly triangularized, that is, there exists an orthogonal matrix $U_\circ$ that simultaneously renders all matrices $U_\circ^\top M_n U_\circ$ upper triangular:

$$\mathrm{low}(U_\circ^\top M_n U_\circ) = 0, \qquad \text{for all} \quad n = 1, \dots N, \tag{3}$$

where $\mathrm{low}(A)$ is the strictly lower triangular part of $A$, i.e., $[\mathrm{low}(A)]_{ij} = A_{ij}$ if $i > j$ and 0 otherwise. On the other hand, when $\sigma > 0$ the observed matrices in $\mathcal{M}_\sigma$ can only be approximately jointly triangularized, for instance by solving the following optimization problem

$$\min_{U \in \mathbb{O}(d)} \mathcal{L}(U), \quad \text{where} \quad \mathcal{L}(U) = \frac{1}{N} \sum_{n=1}^{N} \|\mathrm{low}(U^\top \hat{M}_n U)\|^2, \tag{4}$$

where $\|\cdot\|$ denotes Frobenius norm and optimization is over the manifold $\mathbb{O}(d)$ of orthogonal matrices. The optimization problem can be addressed by Jacobi-like methods [13], or Newton-like methods that optimize directly on the $\mathbb{O}(d)$ manifold [8]. For any feasible $U$ in (4), the joint eigenvalues $\Lambda$ in (1) can be estimated from the diagonals of $U^\top \hat{M}_n U$. This approach has been used in nonsymmetric CP tensor decomposition [22, 8] and other applications [9, 13].

We also note two related problems. In the special case that the ground-truth matrices $M_n$ in $\mathcal{M}_\circ$ are symmetric, the matrix $V$ in (1) is orthogonal, and the estimation problem is known as orthogonal joint diagonalization [7]. Our results apply to this special case too. Another problem is joint diagonalization by congruence [6, 3, 17], in which the matrix $V^{-1}$ in (1) is replaced by $V^T$. In that case the matrix $\Lambda$ in (1) does not contain the joint eigenvalues, and our results do not apply directly.

**Contributions**  We are addressing the joint matrix triangularization problem defined via (4), under the model assumptions (1), (2), and (3). The optimization problem (4) is nonconvex, and hence it is expected to be hard to solve to global optimality in general. Therefore, error bounds are needed that can assess the quality of a solution produced by some algorithm that tries to solve (4). Our main result (Theorem 1) is an error bound that allows to directly assess *a posteriori* the quality of any feasible triangularizer $U$ in (4), in terms of its proximity to the (unknown) exact trangularizer of the ground-truth matrices in $\mathcal{M}_\circ$, regardless of the algorithm used for optimization. The bound depends only on observable quantities and the noise parameter $\sigma$ in (2). The parameter $\sigma$ can often be bounded by a function of the sample size, as in problems involving empirical moment matching [4].

Our approach draws on the perturbation analysis of the Schur decomposition of a single matrix [16]. To our knowledge, our bound in Theorem 1 is the first *a posteriori* error bound for joint matrix decomposition problems. Existing bounds in the literature have a dependence on the ground-truth (and hence unobserved) matrices [11, 17], the proximity of a feasible $U$ to critical points of the objective function [6], or the amount of collinearity between the columns of the matrix $\Lambda$ in (1) [3]. Our error bound is free of such dependencies. Outside the context of joint matrix decomposition, *a posteriori* error bounds have found practical uses in nonconvex optimization [19] and the design of algorithms [21].

**Notation**  All matrices, vectors, and numbers are real. When the context is clear we use 1 to denote the identity matrix. We use $\|\cdot\|$ for matrix Frobenius norm and vector $l_2$ norm. $\mathbb{O}(d)$ is the manifold of orthogonal matrices $U$ such that $U^\top U = 1$. The matrix commutator $[A, B]$ is defined by $[A, B] = AB - BA$. We use $\otimes$ for Kronecker product. For a matrix $A$, we denote by $\lambda_i(A)$ its $i$th eigenvalue, $\lambda_{\min}(A)$ its smallest eigenvalue, $\kappa(A)$ its condition number, $\mathrm{vec}(A)$ its columnwise vectorization, and $\mathrm{low}(A)$ and $\mathrm{up}(A)$ its strictly lower triangular and strictly upper triangular parts, respectively. Low is a binary diagonal matrix defined by $\mathrm{vec}(\mathrm{low}(A)) = \mathrm{Low}\,\mathrm{vec}(A)$. Skew is a skew-symmetric projector defined by $\mathrm{Skew}\,\mathrm{vec}(A) = \mathrm{vec}(A - A^\top)$. $P_{\mathrm{Low}}$ is a $d(d-1)/2 \times d^2$ binary matrix with orthogonal rows, which projects to the subspace of vectorized strictly lower triangular matrices, such that $P_{\mathrm{Low}} P_{\mathrm{Low}}^\top = 1$ and $P_{\mathrm{Low}}^\top P_{\mathrm{Low}} = \mathrm{Low}$. For example, for $d = 3$, one has $\mathrm{Low} = \mathrm{diag}([0, 1, 1, 0, 0, 1, 0, 0, 0])$ and

$$P_{\mathrm{low}} = \begin{pmatrix} 0 & 1 & 0 & 0 & 0 & 0 & 0 & 0 & 0 \\ 0 & 0 & 1 & 0 & 0 & 0 & 0 & 0 & 0 \\ 0 & 0 & 0 & 0 & 0 & 1 & 0 & 0 & 0 \end{pmatrix}.$$

## 2 Perturbation of joint triangularizers

The objective function (4) is continuous in the parameter $\sigma$. This implies that, for $\sigma$ small enough, the approximate joint triangularizers of the observed matrices $\hat{M}_n \in \mathcal{M}_\sigma$ can be expected to be perturbations of the exact triangularizers of the ground-truth matrices $M_n \in \mathcal{M}_\circ$. To formalize this, we express each feasible triangularizer $U$ in (4) as a function of some exact triangularizer $U_\circ$ of the ground-truth matrices, as follows:

$$U = U_\circ e^{\alpha X}, \quad \text{where} \quad X = -X^\top, \quad \|X\| = 1, \quad \alpha > 0, \tag{5}$$

where $e$ denotes matrix exponential and $X$ is a skew-symmetric matrix. Such an expansion holds for any pair $U, U_\circ$ of orthogonal matrices with $\det(U) = \det(U_\circ)$ (see, e.g., [2]). The scalar $\alpha$ in (5) can be interpreted as the 'distance' between the matrices $U$ and $U_\circ$. Our main result is an upper bound on this distance:

**Theorem 1.** *Let $\mathcal{M}_\circ$ and $\mathcal{M}_\sigma$ be the sets of matrices defined in* (1) *and* (2)*, respectively. Let $U$ be a feasible solution of the optimization problem* (4)*, with corresponding value $\mathcal{L}(U)$. Then there exists an orthogonal matrix $U_\circ$ that is an exact joint triangularizer of $\mathcal{M}_\circ$, such that $U$ can be expressed as a perturbation of $U_\circ$ according to* (5)*, with $\alpha$ obeying*

$$\alpha \le \frac{\sqrt{\mathcal{L}(U)} + \sigma}{\sqrt{\lambda_{\min}(\hat{\tau})}} + O(\alpha^2), \qquad \text{where} \tag{6}$$

$$\hat{\tau} = \frac{1}{2N} \sum_{n=1}^N \hat{T}_n^\top \hat{T}_n, \qquad \hat{T}_n = P_{\text{low}} \big(1 \otimes (U^\top \hat{M}_n U) - (U^\top \hat{M}_n^\top U) \otimes 1\big) \text{Skew } P_{\text{low}}^\top. \tag{7}$$

*Proof.* Let $U_\circ$ be the exact joint triangularizer of $\mathcal{M}_\circ$ that is the nearest to $U$ and $\det U = \det U_\circ$. Then $U = U_\circ e^{\alpha X}$ for some unit-norm skew-symmetric matrix $X$ and scalar $\alpha > 0$. Using the expansion $e^{\alpha X} = I + \alpha X + O(\alpha^2)$ and the fact that $X$ is skew-symmetric, we can write, for any $n = 1, \ldots, N$,

$$U^\top \hat{M}_n U = U_\circ^\top \hat{M}_n U_\circ + \alpha[U_\circ^\top \hat{M}_n U_\circ, X] + O(\alpha^2), \tag{8}$$

where $[\cdot, \cdot]$ denotes matrix commutator. Applying the $\text{low}(\cdot)$ operator and using the facts that $\alpha U_\circ^\top \hat{M}_n U_\circ = \alpha U^\top \hat{M}_n U + O(\alpha^2)$ and $\text{low}(U_\circ^\top M_n U_\circ) = 0$, for any $n = 1, \ldots, N$, we can write

$$\alpha \, \text{low}([U^\top \hat{M}_n U, X]) = \text{low}(U^\top \hat{M}_n U) - \sigma \, \text{low}(U_\circ^\top R_n U_\circ) + O(\alpha^2). \tag{9}$$

Stacking (9) over $n$, then taking Frobenius norm, and applying the triangle inequality together with the fact $\|\text{low}(U_\circ^\top R_n U_\circ)\| \le \|U_\circ^\top R_n U_\circ\| = \|R_n\| \le 1$ for all $n = 1, \ldots, N$, we get

$$\alpha \bigg( \sum_{n=1}^N \|\text{low}([U^\top \hat{M}_n U, X])\|^2 \bigg)^{\frac{1}{2}} \le \sqrt{N\mathcal{L}(U)} + \sigma\sqrt{N} + O(\alpha^2), \tag{10}$$

where we used the definition of $\mathcal{L}(U)$ from (4). The rest of the proof involves computing a lower bound of the left-hand side of (10) that holds for all $X$. Since $\|\text{low}(A)\| = \|P_{\text{low}}\text{vec}(A)\|$, we can rewrite the argument of each norm in the left-hand side of (10) as

$$
\begin{aligned}
\text{low}([U^\top \hat{M}_n U, X]) &= P_{\text{low}} \text{vec}([U^\top \hat{M}_n U, X]) \tag{11} \\
&= P_{\text{low}} \big(1 \otimes (U^\top \hat{M}_n U) - (U^\top \hat{M}_n^\top U) \otimes 1\big) \text{vec}(X), \tag{12}
\end{aligned}
$$

and, due to the skew-symmetry of $X$, we can write

$$
\begin{aligned}
\text{vec}(X) &= \text{vec}(\text{low}(X) + \text{up}(X)) = \text{vec}(\text{low}(X) - \text{low}(X)^\top) \tag{13} \\
&= \text{Skew } \text{Low } \text{vec}(X) = \text{Skew } P_{\text{low}}^\top P_{\text{low}} \text{vec}(X). \tag{14}
\end{aligned}
$$

Hence, for all $n = 1, \ldots, N$, we can write $\|\text{low}([U^\top \hat{M}_n U, X])\|^2 = \|\hat{T}_n x\|^2 = x^\top \hat{T}_n^\top \hat{T}_n x$, where $x = P_{\text{low}} \text{vec}(X)$ and $\hat{T}_n$ is defined in (7). The inequality in (6) then follows by using the inequality $x^\top A x \ge \|x\|^2 \lambda_{\min}(A)$, which holds for any symmetric matrix $A$, and noting that $\|x\|^2 = \frac{1}{2}$ (since $x$ contains the lower triangular part of $X$ and $\|X\|^2 = 1$). $\qquad\square$

For general $\mathcal{M}_\sigma$, an analytical expression of $\lambda_{\min}(\hat{\tau})$ in (6) is not available. However, it is straight-forward to compute $\lambda_{\min}(\hat{\tau})$ numerically since all quantities in (7) are observable. Moreover, it is possible to show (see Theorem 2) that in the limit $\sigma \to 0$ and under certain conditions on the ground-truth matrices in $\mathcal{M}_\circ$, the operator $\tau = \lim_{\sigma \to 0} \hat{\tau}$ is nonsingular, i.e., $\lambda_{\min}(\tau) > 0$. Since both $\hat{\tau}$ and $\mathcal{L}$ are continuous in $\sigma$ for $\sigma \to 0$, the boundedness of the right-hand side of (6) is guaranteed, for $\sigma$ small enough, by eigenvalue perturbation theorems.

**Theorem 2.** *The operator $\hat{\tau}$ defined in* (7) *obeys*

$$\lim_{\sigma \to 0} \sqrt{\lambda_{\min}(\hat{\tau})} \geq \frac{\sqrt{\Gamma}}{\kappa(V)^2} , \tag{15}$$

*where $\Gamma = \min_{i > j} \frac{1}{2N} \sum_{n=1}^{N} (\lambda_i(M_n) - \lambda_j(M_n))^2$, and the matrix $V$ is defined in* (1).

The proof is given in the Appendix. The quantity $\Gamma$ can be interpreted as a 'joint eigengap' of $\mathcal{M}_\circ$ (see also [17] for a similar definition in the context of joint diagonalization by congruence). Theorem 2 implies that $\lim_{\sigma \to 0} \lambda_{\min}(\hat{\tau}) > 0$ if $\Gamma > 0$, and the latter is guaranteed under the following condition:

**Condition 1.** *For every $i \neq j$, $i, j = 1, \ldots, d$, there exists at least $n \in \{1, \ldots, N\}$ such that*

$$\lambda_i(M_n) \neq \lambda_j(M_n), \quad where \quad M_n \in \mathcal{M}_\circ . \tag{16}$$

## 3 Experiments

To assess the tightness of the inequality in (6), we created a set of synthetic problems in which the ground truth is known, and we evaluated the bounds obtained from (6) against the true values. Each problem involved the approximate triangularization of a set of randomly generated nearly joint diagonalizable matrices of the form $\hat{M}_n = V \Lambda_n V^{-1} + \sigma R_n$, with $\Lambda_n$ diagonal and $\|R_n\| = 1$, for $n = 1, \ldots, N$. For each set $\mathcal{M}_\sigma = \{\hat{M}_n\}_{n=1}^{N}$, two approximate joint triangularizers were computed by optimizing (4) using two different iterative algorithms, the Gauss-Newton algorithm [8], and the Jacobi algorithm [13] (our implementation), initialized with the same random orthogonal matrix. The obtained solutions $U$ (which may not be the global optima) were then used to compute the empirical bound $\alpha$ from (6), as well as the actual distance parameter $\alpha_{\text{true}} = \|\log U^\top U_\circ\|$, with $U_\circ$ being the global optimum of the unperturbed problem ($\sigma = 0$) that is closest to $U$ and has the same determinant. Locating the closest $U_\circ$ to the given $U$ required checking all $2^d d!$ possible exact triangularizers of $\mathcal{M}_\circ$, thus we restricted our empirical evaluation to the case $d = 5$. We considered two settings, $N = 5$ and $N = 100$, and several different noise levels obtained by varying the perturbation parameter $\sigma$.

The first two graphs in Figure 1 show the value of the noise level $\sigma$ against the values of $\alpha_{\text{true}} = \alpha_{\text{true}}(U)$ and the corresponding empirical bounds $\alpha = \alpha(U)$ from (6), where $U$ are the solutions found by the Gauss-Newton algorithm. (Very similar results were obtained using the Jacobi algorithm.) All values are obtained by averaging over 10 equivalent experiments, and the errorbars show the corresponding standard deviations. For the same set of solutions $U$, the third graph in Figure 1 shows the ratios $\frac{\alpha}{\alpha_{\text{true}}}$.

The experiments show that, at least for small $N$, the bound (6) produces a reasonable estimate of the true perturbation parameter $\alpha_{\text{true}}$. However, our bound does not fully capture the concentration that is expected (and observed in practice) for large sets of nearly jointly decomposable matrices (note, for instance, the average value of $\alpha_{\text{true}}$ in Figure 1, for $N = 5$ vs $N = 100$). This is most likely due to the introduced approximation $\|\text{low}(U_\circ^\top R_n U_\circ)\| \leq 1$ and the use of the triangle inequality in (10) (see proof of Theorem 1), which are needed to separate the observable terms $U^\top \hat{M}_n U$ from the unobservable terms $U_0^\top R_n U_0$ in the right-hand side of (9). Extra assumptions on the distribution of the random matrices $R_n$ can possibly allow obtaining tighter bounds in a probabilistic setting.

## 4 Conclusions

We addressed a joint matrix triangularization problem that involves finding an orthogonal matrix that approximately triangularizes a set of noise-perturbed jointly diagonalizable matrices. The setting can have many applications in statistics and signal processing, in particular in problems that reduce to a nonsymmetric CP tensor decomposition [4, 8, 20]. The joint matrix triangularization problem can be cast as a nonconvex optimization problem over the manifold of orthogonal matrices, and it can be

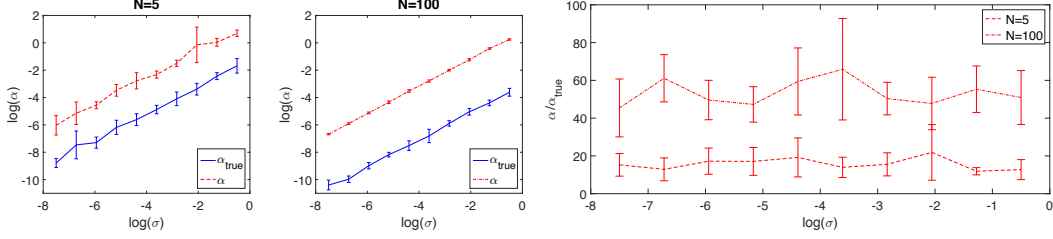

Figure 1: The empirical bound $\alpha$ from (6) vs the true distance $\alpha_{\text{true}}$, on synthetic experiments.

solved numerically but with no success guarantees. We have derived *a posteriori* upper bounds on the distance between any approximate triangularizer (obtained by any algorithm) and the (unknown) solution of the underlying unperturbed problem. The bounds depend only on empirical quantities and hence they can be used to asses the quality of any feasible solution, even when the ground truth is not known. We established that, under certain conditions, the bounds are well defined when the noise is small. Synthetic experiments suggest that the obtained bounds are tight enough to be useful in practice.

In future work, we want to apply our analysis to related problems, such as nonnegative tensor decomposition and simultaneous generalized Schur decomposition [11], and to empirically validate the obtained bounds in machine learning applications [4].

## A    Proof of Theorem 2

The proof consists of two steps. The first step consists of showing that in the limit $\sigma \to 0$ the operator $\hat{\tau}$ defined in (7) tends to a simpler operator, $\tau$, which depends on ground-truth quantities only. The second step is to derive a lower bound on the smallest eigenvalue of the operator $\tau$.

Let $\tau$ be defined by

$$\tau = \frac{1}{2N} \sum_{n=1}^{N} T_n^\top T_n, \qquad T_n = P_{\text{low}}\big(1 \otimes (U_\circ^\top M_n U_\circ) - (U_\circ^\top M_n^\top U_\circ) \otimes 1\big) P_{\text{low}}, \qquad (17)$$

where $M_n \in \mathcal{M}_\circ$, and $U_\circ$ is the exact joint triangularizer of $\mathcal{M}_\circ$ that is closest to, and has the same determinant as, $U$, the approximate joint triangularizer that is used to define $\hat{\tau}$. Proving that $\hat{\tau} \to \tau$ as $\sigma \to 0$ is equivalent to showing that

$$\hat{T}_n\Big|_{\sigma=0} = P_{\text{low}}\big(1 \otimes (U^\top \hat{M}_n U) - (U^\top \hat{M}_n^\top U) \otimes 1\big) \text{Skew } P_{\text{low}}^\top\Big|_{\sigma=0} = T_n. \qquad (18)$$

Since for all $n = 1, \ldots, N$, one has $\hat{M}_n \to M_n$ when $\sigma \to 0$, we need to prove that $U \to U_\circ$ and that we can remove the Skew operator on the right.

We first show that $U = U_\circ e^{\alpha X} \to U_\circ$, that is, $\alpha \to 0$ as $\sigma \to 0$. Assume that the descent algorithm used to obtain $U$ is initialized with $U_{\text{init}}$ obtained from the Schur decomposition of $\hat{M}_* \in \mathcal{M}_\sigma$. Let $U_\circ$ be the exact triangularizer of $\mathcal{M}_\circ$ closest to $U_{\text{init}}$ and $U_{\text{opt}}$ be the local optimum of the joint triangularization objective closest to $U_{\text{init}}$. Then, as $\sigma \to 0$ one has $U_{\text{opt}} \to U_\circ$, by continuity of the objective in $\sigma$, and also $U_{\text{init}} \to U_\circ$ due to the perturbation properties of the Schur decomposition. This implies $U \to U_\circ$, and hence $\alpha \to 0$.

Then, it is easy to prove that $P_{\text{low}}(1 \otimes (U_\circ^\top M_n U_\circ) - (U_\circ^\top M_n^\top U_\circ) \otimes 1) \text{Skew } P_{\text{low}}^\top = P_{\text{low}}(1 \otimes (U_\circ^\top M_n U_\circ) - (U_\circ^\top M_n^\top U_\circ) \otimes 1) P_{\text{low}}^\top$ by considering the action of the two operators on $x = P_{\text{low}}\text{vec}(X)$, with $X = -X^\top$. One has

$$P_{\text{low}}\big(1 \otimes (U_\circ^\top M_n U_\circ) - (U_\circ^\top M_n^\top U_\circ) \otimes 1\big) \text{Skew } P_{\text{low}}^\top = P_{\text{low}}^\top \text{vec}\big(\text{low}[U_\circ^\top M_n U_\circ, X]\big)$$
$$= P_{\text{low}}^\top \text{vec}\big(\text{low}[U_\circ^\top M_n U_\circ, \text{low}(X)]\big) = P_{\text{low}}(1 \otimes (U_\circ^\top M_n U_\circ) - (U_\circ^\top M_n^\top U_\circ) \otimes 1) P_{\text{low}}^\top \quad (19)$$

where in the second line we used the fact that $U_\circ^\top M_n^\top U_\circ$ is upper triangular. This shows that $\hat{\tau} \to \tau$ as $\sigma \to 0$.

The second part of the proof consists of bounding the smallest eigenvalue of $\tau$. We will make use of the following identity that holds when $A$ and $C$ are upper triangular:

$$\text{low}(ABC) = \text{low}(A\,\text{low}(B)\,C)\,, \tag{20}$$

from which we get the following identity when $A$ and $C$ are upper triangular:

$$\text{Low vec}(ABC) = \text{Low}\,(C^\top \otimes A)\,\text{Low vec}(B)\,. \tag{21}$$

In particular, one has $P_{\text{low}}^\top T_n x = \text{Lowvec}([U_\circ^\top M_n U_\circ, \text{low}(X)]) = \text{Lowvec}(U_\circ^\top M_n U_\circ \text{low}(X) - \text{low}(X)U_\circ^\top M_n U_\circ)$ and it can be shown that[1]

$$\text{Low vec}(\tilde{V}\Lambda_n \tilde{V}^{-1}\text{low}(X)) = \text{Low}\,(\tilde{V}^{-T} \otimes \tilde{V})\,\text{Low}\,(I \otimes \Lambda_n)\,\text{Low}\,(\tilde{V}^\top \otimes \tilde{V}^{-1})\,\text{Low vec}(X) \tag{29}$$

and

$$\text{Low vec}(\text{low}(X)\tilde{V}\Lambda_n \tilde{V}^{-1}) = \text{Low}\,(\tilde{V}^{-T} \otimes \tilde{V})\,\text{Low}\,(\Lambda_n \otimes I)\,\text{Low}\,(\tilde{V}^\top \otimes \tilde{V}^{-1})\,\text{Low vec}(X) \tag{30}$$

where $\tilde{V}$ and $\tilde{V}^{-1}$ are upper triangular matrices defined by $U_\circ^\top M_n U_\circ = \tilde{V}\Lambda_n \tilde{V}^{-1}$. Now, since $\tilde{V} = U_\circ^\top V$, where $V$ is defined via the spectral decomposition $M_n = V\Lambda_n V^{-1}$, we can rewrite the operator $T_n$ as

$$T_n = P_{\text{low}}(U_\circ^\top V^{-T} \otimes U_\circ^\top V)\text{Low}(1 \otimes \Lambda_n - \Lambda_n \otimes 1)\text{Low}(V^T U_\circ \otimes V^{-1}U_\circ)P_{\text{low}}^\top\,, \tag{31}$$

and use the following inequality for the smallest eigenvalue of $\tau = \frac{1}{2N}\sum_{n=1}^N T_n^\top T_n$:

$$\lambda_{\min}(\tau) \geq \frac{1}{2N}\lambda_{\min}(A)\,\lambda_{\min}(B)\,\lambda_{\min}(C), \tag{32}$$

where

$$A = P_{\text{low}}(U_\circ^\top V \otimes U_\circ^\top V^{-T})P_{\text{low}}^\top, \tag{33}$$

$$B = P_{\text{low}}\left(\sum_{n=1}^N (1 \otimes \Lambda_n - \Lambda_n \otimes 1)^2\right)P_{\text{low}}^\top, \tag{34}$$

$$C = P_{\text{low}}(V^{-1}U_\circ \otimes V^T U_\circ)P_{\text{low}}^\top. \tag{35}$$

Now, it is easy to show that $\lambda_{\min}(A) = \lambda_{\min}(C) \geq \frac{1}{\kappa(V)^2}$ since the $d(d-1)/2 \times d(d-1)/2$ matrices $A$ and $C$ are obtained by deleting certain rows and columns of $U_\circ^\top V \otimes U_\circ^\top V^{-T}$ and $V^{-1}U_\circ \otimes V^T U_\circ$ respectively. The matrix $B$ is a diagonal matrix with entries given by

$$[B]_{kk} = \sum_{n=1}^N ([\Lambda_n]_{ii} - [\Lambda_n]_{jj})^2, \qquad k = j - i + \sum_{a=1}^{i-1}(d - a), \tag{36}$$

with $0 < i < j$ and $j = 1,\ldots d$. This implies $\lambda_{\min}(B) = \min_{i<j}\sum_{n=1}^N (\lambda_i(M_n) - \lambda_j(M_n))^2$ and

$$\lim_{\sigma \to 0}\lambda_{\min}(\hat{\tau}) \geq \frac{\Gamma}{\kappa(V)^4}\,, \tag{37}$$

where $\Gamma = \min_{i>j}\frac{1}{2N}\sum_{n=1}^N (\lambda_i(M_n) - \lambda_j(M_n))^2$ is a 'joint eigengap' of the ground-truth matrices $M_n \in \mathcal{M}_\circ$. $\qquad\square$

$$\text{Low vec}(\tilde{V}\Lambda_n \tilde{V}^{-1}Y) = \tag{22}$$

$$\text{Low vec}(\tilde{V}\Lambda_n \tilde{V}^{-1}Y\tilde{V}\tilde{V}^{-1}) = \tag{23}$$

$$\text{Low}\,(\tilde{V}^{-T} \otimes \tilde{V})\,\text{Low vec}(\Lambda_n \tilde{V}^{-1}Y\tilde{V}) = \tag{24}$$

$$\text{Low}\,(\tilde{V}^{-T} \otimes \tilde{V})\,\text{Low}\,(I \otimes \Lambda_n)\,\text{Low vec}(\tilde{V}^{-1}Y\tilde{V}) = \tag{25}$$

$$\text{Low}\,(\tilde{V}^{-T} \otimes \tilde{V})\,\text{Low}\,(I \otimes \Lambda_n)\,\text{Low}\,(\tilde{V}^\top \otimes \tilde{V}^{-1})\,\text{Low vec}(Y) \tag{26}$$

and similarly

$$\text{Low vec}(Y\tilde{V}\Lambda_n \tilde{V}^{-1}) = \tag{27}$$

$$\text{Low}\,(\tilde{V}^{-T} \otimes \tilde{V})\,\text{Low}\,(\Lambda_n \otimes I)\,\text{Low}\,(\tilde{V}^\top \otimes \tilde{V}^{-1})\,\text{Low vec}(Y) \tag{28}$$

## Footnotes

[1]For any matrix Y one has

# References

[1] K. Abed-Meraim and Y. Hua. A least-squares approach to joint Schur decomposition. In *Acoustics, Speech and Signal Processing, 1998. Proceedings of the 1998 IEEE International Conference on*, volume 4, pages 2541–2544. IEEE, 1998.

[2] P.-A. Absil, R. Mahony, and R. Sepulchre. *Optimization algorithms on matrix manifolds*. Princeton University Press, 2009.

[3] B. Afsari. Sensitivity analysis for the problem of matrix joint diagonalization. *SIAM Journal on Matrix Analysis and Applications*, 30(3):1148–1171, 2008.

[4] A. Anandkumar, R. Ge, D. Hsu, S. Kakade, and M. Telgarsky. Tensor decompositions for learning latent variable models. *Journal of Machine Learning Research*, 15:2773–2832, 2014.

[5] B. Balle, A. Quattoni, and X. Carreras. A spectral learning algorithm for finite state transducers. In *Machine Learning and Knowledge Discovery in Databases*, pages 156–171. Springer, 2011.

[6] J.-F. Cardoso. Perturbation of joint diagonalizers. *Telecom Paris, Signal Department, Technical Report 94D023*, 1994.

[7] J.-F. Cardoso and A. Souloumiac. Jacobi angles for simultaneous diagonalization. *SIAM journal on matrix analysis and applications*, 17(1):161–164, 1996.

[8] N. Colombo and N. Vlassis. Tensor decomposition via joint matrix Schur decomposition. In *Proc. 33rd International Conference on Machine Learning*, 2016.

[9] R. M. Corless, P. M. Gianni, and B. M. Trager. A reordered Schur factorization method for zero-dimensional polynomial systems with multiple roots. In *Proceedings of the 1997 international symposium on Symbolic and algebraic computation*, pages 133–140. ACM, 1997.

[10] L. De Lathauwer. A link between the canonical decomposition in multilinear algebra and simultaneous matrix diagonalization. *SIAM Journal on Matrix Analysis and Applications*, 28(3):642–666, 2006.

[11] L. De Lathauwer, B. De Moor, and J. Vandewalle. Computation of the canonical decomposition by means of a simultaneous generalized Schur decomposition. *SIAM Journal on Matrix Analysis and Applications*, 26(2):295–327, 2004.

[12] T. Fu, S. Jin, and X. Gao. Balanced simultaneous Schur decomposition for joint eigenvalue estimation. In *Communications, Circuits and Systems Proceedings, 2006 International Conference on*, volume 1, pages 356–360. IEEE, 2006.

[13] M. Haardt and J. A. Nossek. Simultaneous Schur decomposition of several nonsymmetric matrices to achieve automatic pairing in multidimensional harmonic retrieval problems. *Signal Processing, IEEE Transactions on*, 46(1):161–169, 1998.

[14] R. A. Horn and C. R. Johnson. *Matrix analysis*. Cambridge University Press, 2nd edition, 2012.

[15] R. Iferroudjene, K. A. Meraim, and A. Belouchrani. A new Jacobi-like method for joint diagonalization of arbitrary non-defective matrices. *Applied Mathematics and Computation*, 211(2):363–373, 2009.

[16] M. Konstantinov, P. H. Petkov, and N. Christov. Nonlocal perturbation analysis of the Schur system of a matrix. *SIAM Journal on Matrix Analysis and Applications*, 15(2):383–392, 1994.

[17] V. Kuleshov, A. Chaganty, and P. Liang. Tensor factorization via matrix factorization. In *18th International Conference on Artificial Intelligence and Statistics (AISTATS)*, 2015.

[18] X. Luciani and L. Albera. Joint eigenvalue decomposition using polar matrix factorization. In *International Conference on Latent Variable Analysis and Signal Separation*, pages 555–562. Springer, 2010.

[19] J.-S. Pang. A posteriori error bounds for the linearly-constrained variational inequality problem. *Mathematics of Operations Research*, 12(3):474–484, 1987.

[20] A. Podosinnikova, F. Bach, and S. Lacoste-Julien. Beyond CCA: Moment matching for multi-view models. In *Proc. 33rd International Conference on Machine Learning*, 2016.

[21] S. Prudhomme, J. T. Oden, T. Westermann, J. Bass, and M. E. Botkin. Practical methods for a posteriori error estimation in engineering applications. *International Journal for Numerical Methods in Engineering*, 56(8):1193–1224, 2003.

[22] S. H. Sardouie, L. Albera, M. B. Shamsollahi, and I. Merlet. Canonical polyadic decomposition of complex-valued multi-way arrays based on simultaneous Schur decomposition. In *Acoustics, Speech and Signal Processing (ICASSP), 2013 IEEE International Conference on*, pages 4178–4182. IEEE, 2013.

[23] A. Souloumiac. Nonorthogonal joint diagonalization by combining Givens and hyperbolic rotations. *IEEE Transactions on Signal Processing*, 57(6):2222–2231, 2009.

